# Phoneme Recognition with Large Hierarchical Reservoirs

**Fabian Triefenbach**    **Azarakhsh Jalalvand**    **Benjamin Schrauwen**

**Jean-Pierre Martens**

Department of Electronics and Information Systems
Ghent University
Sint-Pietersnieuwstraat 41, 9000 Gent, Belgium
`fabian.triefenbach@elis.ugent.be`

## Abstract

Automatic speech recognition has gradually improved over the years, but the reliable recognition of unconstrained speech is still not within reach. In order to achieve a breakthrough, many research groups are now investigating new methodologies that have potential to outperform the Hidden Markov Model technology that is at the core of all present commercial systems. In this paper, it is shown that the recently introduced concept of Reservoir Computing might form the basis of such a methodology. In a limited amount of time, a reservoir system that can recognize the elementary sounds of continuous speech has been built. The system already achieves a state-of-the-art performance, and there is evidence that the margin for further improvements is still significant.

## 1 Introduction

Thanks to a sustained world-wide effort, modern automatic speech recognition technology has now reached a level of performance that makes it suitable as an enabling technology for novel applications such as automated dictation, speech based car navigation, multimedia information retrieval, etc. Basically all state-of-the-art systems utilize Hidden Markov Models (HMMs) to compose an acoustic model that captures the relations between the acoustic signal and the phonemes, defined as the basic contrastive units of the sound system of a spoken language. The HMM theory has not changed that much over the years, and the performance growth is slow and for a large part owed to the availability of more training data and computing resources.

Many researchers advocate the need for alternative learning methodologies that can supplement or even totally replace the present HMM methodology. In the nineties for instance, very promising results were obtained with Recurrent Neural Networks (RNNs) [1] and hybrid systems both comprising neural networks and HMMs [2], but these systems were more or less abandoned since then. More recently, there was a renewed interest in applying new results originating from the Machine Learning community. Two techniques, namely Deep Belief Networks (DBNs) [3, 4] and Long Short-Term Memory (LSTM) recurrent neural networks [5], have already been used with great success for phoneme recognition. In this paper we present the first (to our knowledge) phoneme recognizer that employs Reservoir Computing (RC) [6, 7, 8] as its core technology.

The basic idea of Reservoir Computing (RC) is that complex classifications can be performed by means of a set of simple linear units that 'read-out' the outputs of a pool of fixed (not trained) nonlinear interacting neurons. The RC concept has already been successfully applied to time series generation [6], robot navigation [9], signal classification [8], audio prediction [10] and isolated

spoken digit recognition [11, 12, 13]. In this contribution we envisage a RC system that can recognize the English phonemes in continuous speech. In a short period (a couple of months) we have been able to design a hierarchical system of large reservoirs that can already compete with many state-of-the-art HMMs that have only emerged after several decades of research.

The rest of this paper is organized as follows: in Section 2 we describe the speech corpus we are going to work on, in Section 3 we recall the basic principles of Reservoir Computing, in Section 4 we discuss the architecture of the reservoir system which we propose for performing Large Vocabulary Continuous Speech Recognition (LVCSR), and in Section 5 we demonstrate the potential of this architecture for phoneme recognition.

## 2   The speech corpus

Since the main aim of this paper is to demonstrate that reservoir computing can yield a good acoustic model, we will conduct experiments on TIMIT, an internationally renowned corpus [14] that was specifically designed to support the development and evaluation of such a model.

The TIMIT corpus contains 5040 English sentences spoken by 630 different speakers representing eight dialect groups. About 70% of the speakers are male, the others are female. The corpus documentation defines a training set of 462 speakers and a test set of 168 different speakers: a main test set of 144 speakers and a core test set of 24 speakers. Each speaker has uttered 10 sentences: two SA sentences which are the same for all speakers, 5 SX-sentences from a list of 450 sentences (each one thus appearing 7 times in the corpus) and 3 SI-sentences from a set of 1890 sentences (each one thus appearing only once in the corpus). To avoid a biased result, the SA sentences will be excluded from training and testing.

For each utterance there is a manual acoustic-phonetic segmentation. It indicates where the phones, defined as the atomic units of the acoustic realizations of the phonemes, begin and end. There are 61 distinct phones, which, for evaluation purposes, are usually reduced to an inventory of 39 symbols, as proposed by [15]. Two types of error rates can be reported for the TIMIT corpus. One is the Classification Error Rate (CER), defined as the percentage of the time the top hypothesis of the tested acoustic model is correct. The second one is the Recognition Error Rate (RER), defined as the ratio between the number of edit operations needed to convert the recognized symbol sequence into the reference sequence, and the number of symbols in that reference sequence. The edit operations are symbol deletions, insertions and substitutions. Both classification and recognition can be performed at the phone and the phoneme level.

## 3   The basics of Reservoir Computing

In this paper, a Reservoir Computing network (see Figure 1) is an Echo State Network [6, 7, 8] consisting of a fixed dynamical system (the reservoir) composed of nonlinear recurrently connected neurons which are left untrained, and a set of linear output nodes (read-out nodes). Each output node is trained to recognize one class (one-vs-all classification). The number of connections between and within layers can be varied from sparsely connected to fully connected. The reservoir neurons have an activation function f(x) = logistic($x$).

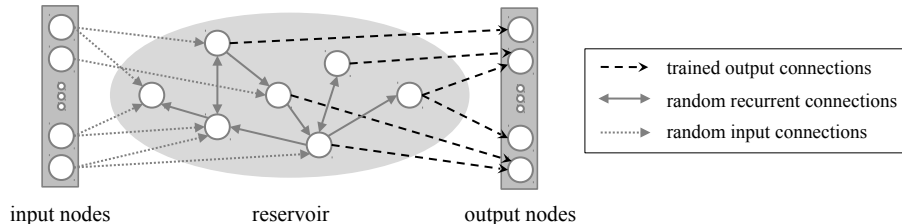

Figure 1: A reservoir computing network consists of a reservoir of fixed recurrently connected nonlinear neurons which are stimulated by the inputs, and an output layer of trainable linear units.

The RC approach avoids the back-propagation through time learning which can be very time consuming and which suffers from the problem of vanishing gradients [6]. Instead, it employs a simple and efficient linear regression learning of the output weights. The latter tries to minimize the mean squared error between the computed and the desired outputs at all time steps.

Based on its recurrent connections, the reservoir can capture the long-term dynamics of the human articulatory system to perform speech sound classification. This property should give it an advantage over HMMs that rely on the assumption that subsequent acoustical input vectors are conditionally independent.

Besides the 'memory' introduced through the recurrent connections, the neurons themselves can also integrate information over time. Typical neurons that can accomplish this are Leaky Integrator Neurons (LINs) [16]. With such neurons the reservoir state at time $k+1$ can be computed as follows:

$$\mathbf{x}[k+1] = (1 - \lambda)\mathbf{x}[k] + \lambda f(\mathbf{W}_{res}\mathbf{x}[k] + \mathbf{W}_{in}\mathbf{u}[k]) \tag{1}$$

with $\mathbf{u}[k]$ and $\mathbf{x}[k]$ representing the inputs and the reservoir state at time $k$. The $\mathbf{W}$ matrices contain the input and recurrent connection weights. It is common to include a constant bias in $\mathbf{u}[k]$. As long as the leak rate $\lambda < 1$, the integration function provides an additional fading memory of the reservoir state.

To perform a classification task, the RC network computes the outputs at time $k$ by means of the following linear equation:

$$\mathbf{y}[k] = \mathbf{W}_{out}\,\mathbf{x}[k] \tag{2}$$

The reservoir state in this equation is augmented with a constant bias. If the reservoir states at the different time instants form the columns of a large state matrix $\mathbf{X}$ and if the corresponding desired outputs form the columns of a matrix $\mathbf{D}$, the optimal $\mathbf{W}_{out}$ emerges from the following equations:

$$\mathbf{W}_{out} = \arg\min_{\mathbf{W}} \left( \frac{1}{N} \left( ||\mathbf{X}\,\mathbf{W} - \mathbf{D}||^2 + \epsilon\,||\mathbf{W}||^2 \right) \right) \tag{3}$$

$$\mathbf{W}_{out} = (\mathbf{X^T X} + \epsilon\,\mathbf{I})^{-1}(\mathbf{X^T D}) \tag{4}$$

with N being the number of frames. The regularization constant $\epsilon$ aims to limit the norm of the output weights (this is the so-called Tikhonov or ridge regression). For large training sets, as common in speech processing, the matrices $\mathbf{X^T X}$ and $\mathbf{X^T D}$ are updated on-line in order to suppress the need for huge storage capacity. In this paper, the regularization parameter $\epsilon$ was fixed to $10^{-8}$. This regularization is equivalent to adding Gaussian noise with a variance of $10^{-8}$ to the reservoir state variables.

## 4  System architecture

The main objective of our research is to build an RC-based LVCSR system that can retrieve the words from a spoken utterance. The general architecture we propose for such a system is depicted in Figure 2. The preprocessing stage converts the speech waveform into a sequence of acoustic

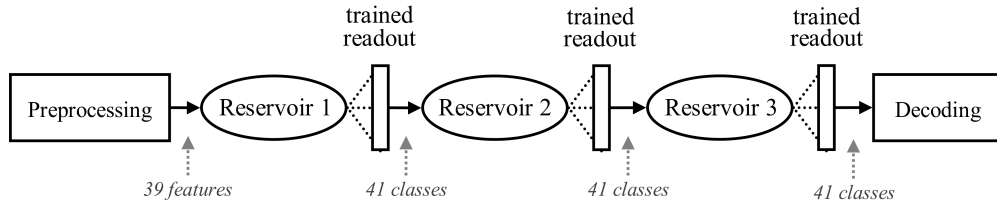

Figure 2: Hierarchical reservoir architecture with multiple layers.

feature vectors representing the acoustic properties in subsequent speech frames. This sequence is supplied to a hierarchical system of RC networks. Each reservoir is composed of LINs which are fully connected to the inputs and to the 41 outputs. The latter represent the distinct phonemes of the language. The outputs of the last RC network are supplied to a decoder which retrieves the most likely linguistic interpretation of the speech input, given the information computed by the RC

networks and given some prior knowledge of the spoken language. In this paper, the decoder is a phoneme recognizer just accommodating a bigram phoneme language model. In a later stage it will be extended with other components: (1) a phonetic dictionary comprising all the words of the system's vocabulary and their common pronunciations, expressed as phoneme sequences, and (2) a n-gram language model describing the probabilities of each word, given the preceding (n-1) words.

We conjecture that the integration time of the LINs in the first reservoir should ideally be long enough to capture the co-articulations between successive phonemes emerging from the dynamical constraints of the articulatory system. On the other hand, it has to remain short enough to avoid that information pointing to the presence of a short phoneme is too much blurred by the left phonetic context. Furthermore, we argue that additional reservoirs can correct some of the errors made by the first reservoir. Indeed, such an error correcting reservoir can guess the correct labels from its inputs, and take the past phonetic context into account in an implicit way to refine the decision. This is in contrast to an HMM system which adopts an explicit approach, involving separate models for several thousands of context-dependent phonemes.

In the next subsections we provide more details about the different parts of our recognizer, and we also discuss the tuning of some of its control parameters.

## 4.1 Preprocessing

The preprocessor utilizes the standard Mel Frequency Cepstral Coefficient (MFCC) analysis [17] encountered in most state-of-the-art LVCSR systems. The analysis is performed on 25 ms Hamming-windowed speech frames, and subsequent speech frames are shifted over 10 ms with respect to each other. Every 10 ms a 39-dimensional feature vector is generated. It consists of 13 static parameters, namely the log-energy and the first 12 MFCC coefficients, their first order derivatives (the velocity or $\Delta$ parameters), and their second order derivatives (the acceleration or $\Delta\Delta$ parameters).

In HMM systems, the training is insensitive to a linear rescaling of the individual features. In RC systems however, the input and recurrent weights are not trained and drawn from predefined statistical distributions. Consequently, by rescaling the features, the impact of the inputs on the activations of the reservoir neurons is changed as well, which makes it compulsory to employ an appropriate input scaling [8].

To establish a proper input scaling the acoustic feature vector is split into six sub-vectors according to the dimensions (energy, cepstrum) and (static, velocity, acceleration). Then, each feature $a_i$, $(i = 1, .., 39)$ is normalized to $z_i = \alpha_s (u_i - \overline{u}_i)$ with $\overline{u}_i$ being the mean of $u_i$ and $s$ $(s = 1, .., 6)$ referring to the sub-vector (group) the feature belongs to. The aim of $\alpha_s$ is to ensure that the norm of each sub-vector is one. If the $z_i$ were supplied to the reservoir, each sub-vector would on average have the same impact on the reservoir neuron activations. Therefore, in a second stage, the $z_i$ are rescaled to $u_i = \beta_s z_i$ with $\beta_s$ representing the relative importance of sub-vector $s$ in the reservoir neuron activations. The normalization constants $\alpha_s$ straightly follow from a statistical analysis of the

Table 1: Different types of acoustic information in the input features and their optimal scale factors.

| group name | Energy features | | | Cepstral features | | |
| --- | --- | --- | --- | --- | --- | --- |
| | $\log(\mathbf{E})$ | $\Delta \log(\mathbf{E})$ | $\Delta\Delta \log(\mathbf{E})$ | $c_{1...12}$ | $\Delta c_{1...12}$ | $\Delta\Delta c_{1...12}$ |
| norm factor $\alpha$ | 0.27 | 1.77 | 4.97 | 0.10 | 0.61 | 1.75 |
| scale factor $\beta$ | 1.75 | 1.25 | 1.00 | 1.25 | 0.50 | 0.25 |

acoustic feature vectors. The factors $\beta_s$ are free parameters that were selected such that the phoneme classification error of a single reservoir system of 1000 neurons is minimized on the validation set. The obtained factors (see Table 1) confirm that the static features are more important than the velocity and the acceleration features.

The proposed rescaling has the following advantages: it preserves the relative importance of the individual features within a sub-vector, it is fully defined by six scaling parameters $\alpha_s \beta_s$, it takes only a minimal computational effort, and it is actually supposed to work well for any speech corpus.

## 4.2 Sequence decoding

The decoder in our present system performs a Viterbi search for the most likely phonemic sequence given the acoustic inputs and a bigram phoneme language model. The search is driven by a simple model for the conditional likelihood $p(\mathbf{y}|m)$ that the reservoir output vector $\mathbf{y}$ is observed during the acoustical realization of phoneme $m$. The model is based on the cosine similarity between $\mathbf{y} + \mathbf{1}$ and a template vector $\mathbf{t}_m = [0, .., 0, 1, 0, .., 0]$, with its nonzero element appearing at position $m$. Since the template vector is a unity vector, we compute $p(\mathbf{y}|m)$ as

$$p(\mathbf{y}|m) = \left( \max[0, \frac{< \mathbf{y} + \mathbf{1}, \mathbf{t}_m >)}{\sqrt{< \mathbf{y} + \mathbf{1}, \mathbf{y} + \mathbf{1} >}}] \right)^{\kappa}, \qquad (5)$$

with $< \mathbf{x}, \mathbf{y} >$ denoting the dot product of vectors $\mathbf{x}$ and $\mathbf{y}$. Due to the offset, we can ensure that the components of $\mathbf{y} + \mathbf{1}$ are between 0 and 1 most of the time. The maximum operator prevents the likelihoods from becoming negative occasionally. The exponent $\kappa$ is a free parameters that will be tuned experimentally. It controls the relative importance of the acoustic model and the bigram phoneme language model.

## 4.3 Reservoir optimization

The training of the reservoir output nodes is based on Equations (3) and (4) and the desired phoneme labels emerge from a time synchronized phonemic transcription. The latter was derived from the available acoustic-phonetic segmentation of TIMIT. For all experiments reported in this paper, we have used the modular RC toolkit OGER[1] developed at Ghent University.

The recurrent weights of the reservoir are not trained but randomly drawn from statistical distributions. The input weights emerge from a uniform distribution between $-U$ and $+U$, the recurrent weights from a zero-mean Gaussian distribution with a variance $V$. The value of $U$ controls the relative importance of the inputs in the activation of the reservoir neurons and is often called the input scale factor (ISF). The variance $V$ directly determines the spectral radius (SR), defined as the largest absolute eigenvalue of the recurrent weight matrix. The SR describes the dynamical excitability of the reservoir [6, 8]. The SR and the ISF must be jointly optimized. To do so, we used 1000 neuron reservoirs, supplied with inputs that were normalized according to the procedure reviewed in the previous section. We found that SR = 0.4 and ISF = 0.4 yield the best performance, but for SR $\in (0.3...0.8)$ and for ISF $\in (0.2...1.0)$, the performance is quite stable.

Another parameter that must be optimized is the leak rate, denoted as $\lambda$. It determines the integration time of the neurons. If the nonlinear function is ignored and the time between frames is $T_f$, the reservoir neurons represent a first-order leaky integrator with a time constant $\tau$ that is related to $\lambda$ by $\lambda = 1 - e^{-T_f/\tau}$. As stated before, the integration time should be long enough to capture the relevant co-articulation effects and short enough to constrain the information blurring over subsequent phonemes. This is confirmed by Figure 3 showing how the phoneme CER of a single reservoir system changes as a function of the integrator time constant. The optimal value is 40 ms,

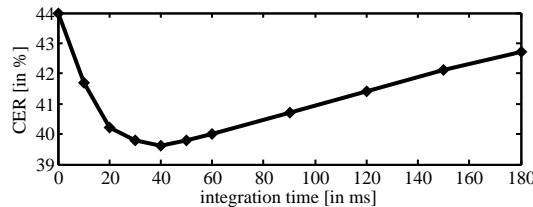

Figure 3: The phoneme Classification Error Rate (CER) as a function of the integration time (in ms)

and completely in line with psychophysical data concerning the post and pre-masking properties of the human auditory system. In [18] for instance, it is shown that these properties can be explained by means of a second order low-pass filter with real poles corresponding to time constants of 8 and 40 ms respectively (it is the largest constant that determines the integration time here).

It has been reported [19] that one can easily reduce the number of recurrent connections in a RC network without much affecting its performance. We have found that limiting the number of connection to 50 per neuron does not harm the performance while it dramatically reduces the required computational resources (memory and computation time).

# 5 Experiments

Since our ultimate goal is to perform LVCSR, and since LVCSR systems work with a dictionary of phonemic transcriptions, we have worked with phonemes rather than with phones. As in [20] we consider the 41 phoneme symbols one encounters in a typical phonetic dictionary like COMLEX [21]. The 41 symbols are very similar to the 39 symbols of the reduced phone set proposed by [15], but with one major difference, namely, that a phoneme string does not contain any silences referring to closures of plosive sounds (e.g. the closure */kcl/* of phoneme */k/*). By ignoring confusions between */sh/* and */zh/* and between */ao/* and */aa/* we finally measure phoneme error rates for 39 classes, in order to make them more compliant with the phone error rates for 39 classes reported in other papers. Nevertheless, we will see later that phoneme recognition is harder to accomplish than phone recognition. This is because the closures are easy to recognize and contribute to a low phone error rate. In phoneme recognition there are no closure anymore.

In what follows, all parameter tuning is performed on the TIMIT training set (divided into independent training and development sets), and all error rates are measured on the main test set. The bigram phoneme language model used for the sequence decoding step is created from the phonemic transcriptions of the training utterances.

## 5.1 Single reservoir systems

In a first experiment we assess the performance of a single reservoir system as a function of the reservoir size, defined as the number of neurons in the reservoir. The phoneme targets during training are derived from the manual acoustic-phonetic segmentation, as explained in Section 4.3. We increase the number of neurons from 500 to 20000. The corresponding number of trainable parameters then changes from 20K to 800K. The latter figure corresponds to the number of trainable parameters in an HMM system comprising 1200 independent Gaussian mixture distributions of 8 mixtures each.

Figure 4 shows that the phoneme CER on the training set drops by about 4% every time the reservoir size is doubled. The phoneme CER on the test set shows a similar trend, but the slope is decreasing from 4% at low reservoir sizes to 2% at 20000 neurons (nodes). At that point the CER on the test

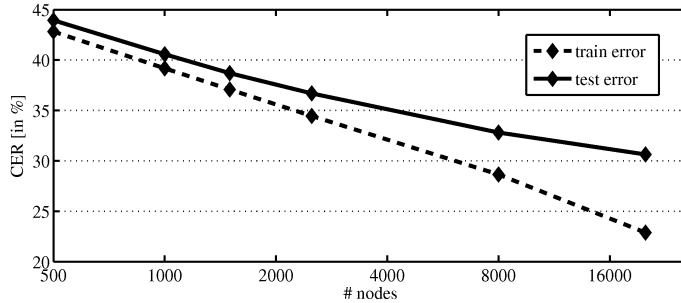

Figure 4: The Classification Error Rate (CER) at the phoneme level for the training and test set as a function of the reservoir size.

set is 30.6% and the corresponding RER (not shown) is 31.4%. The difference between the test and the training error is about 8%.

Although the figures show that an even larger reservoir will perform better, we stopped at 20000 nodes because the storage and the inversion of the large matrix $\mathbf{X^T X}$ are getting problematic. Before starting to investigate even larger reservoirs, we first want to verify our hypothesis that adding a second (equally large) layer can lead to a better performance.

## 5.2 Multilayer reservoir systems

Usually, a single reservoir system produces a number of competing outputs at all time steps, and this hampers the identification of the correct phoneme sequence. The left panel of Figure 5 shows the outputs of a reservoir of 8000 nodes in a time interval of 350 ms. Our hypothesis was that the observed confusions are not arbitrary, and that a second reservoir operating on the outputs of the first reservoir system may be able to discover regularities in the error patterns. And indeed, the outputs of this second reservoir happen to exhibit a larger margin between the winner and the competition, as illustrated in the right panel of Figure 5.

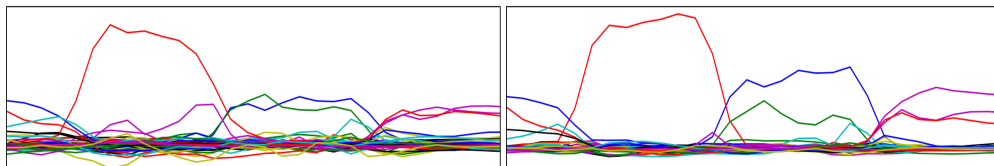

Figure 5: The outputs of the first (left) and the second (right) layer of a two-layer system composed of two 8000 node reservoirs. The shown interval is 350 ms long.

In Figure 6, we have plotted the phoneme CER and RER as a function of the number of reservoirs (layers) and the size of these reservoirs. We have thus far only tested systems with equally large reservoirs at every layer. For the exponent $\kappa$, we have just tried $\kappa = 0.5$, 0.7 and 1, and we have selected the value yielding the best balance between insertions and deletions.

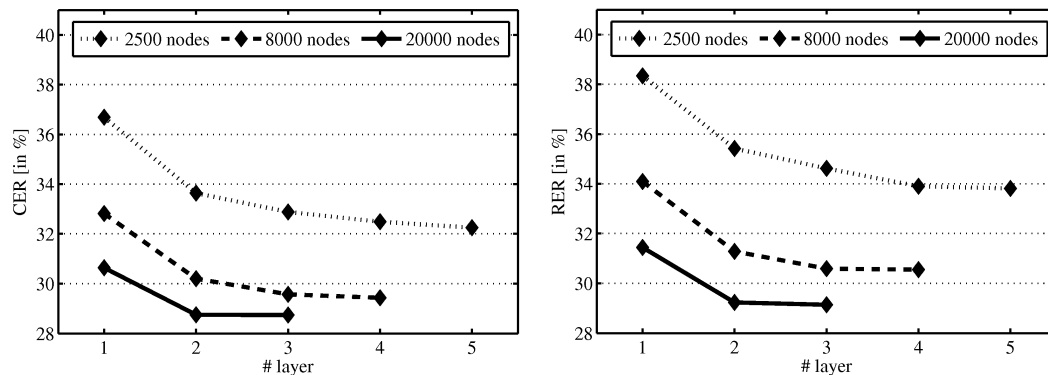

Figure 6: The phoneme CERs and RERs for different combinations of number of nodes and layers

For all reservoir sizes, the second layer induces a significant improvement of the CER by 3-4% absolute. The corresponding improvements of the recognition error rates are a little bit less but still significant. The best RER obtained with a two-layer system comprising reservoirs of 20000 nodes is 29.1%. Both plots demonstrate that a third layer does not cause any additional gain when the reservoir size is large enough. However, this might also be caused by the fact that we did not systematically optimize the parameters (SR, leak rate, regularization parameter, etc.) for each large system configuration we investigated. We just chose sensible values which were retrieved from tests with smaller systems.

## 5.3 Comparison with the state-of-the-art

In Table 2 we have listed some published results obtained on TIMIT with state-of-the-art HMM systems and other recently proposed research systems. We have also included the results of own experiments we conducted with SPRAAK[2] [22], a recently launched HMM-based speech recognition toolkit. In order to provide an easier comparison, we also build a phone recognition system based on the same design parameters that were optimized for phoneme recognition. All phone RERs are

calculated on the core test set, while the phoneme RERs were measured on the main test set. We do this because most figures in speech community papers apply to these experimental settings. Our final results were obtained with systems that were trained on the full training data (including the development set). Before discussing our figures in detail we emphasize that the two figures for SPRAAK confirm our earlier statement that phoneme recognition is harder than phone recognition.

Table 2: Phoneme and Phone Recognition Error Rates (in %) obtained with state-of-the-art systems.

| System description<br>used test set | Phone RER<br>core test | Phoneme RER<br>main test |
|---|---|---|
| **Reservoir Computing** (this paper) | **26.8** | **29.1** |
| CD-HMM (SPRAAK Toolkit) | 25.6 | 28.1 |
| CD-HMM [20] | | 28.7 |
| Recurrent Neural Networks [1] | 26.1 | |
| LSTM+CTC [5] | (24.6) | |
| Bayesian Triphone HMM [23] | 24.4 | |
| Deep Belief Networks [4] | 23.0 | |
| Hierarchical HMM + MLPs [20] | | (23.4) |

Given the fact that SPRAAK seems to achieve state-of-the-art performance, it is fair to conclude from the figures in Table 2 that our present system is already competitive with other modern HMM systems. It is also fair to say that better systems do exist, like the Deep Belief Network system [4] and the hierarchical HMM system with multiple Multi-Layer Perceptrons (MLPs) on top of an HMM system [20]. Note however that the latter system also employs complex temporal patterns (TRAPs) as input features. These patterns are much more powerful than the simple MFCC vectors used in all other systems we cite. Furthermore, the LSTM+CTC [5] results too must be considered with some care since they were obtained with a bidirectional system. Such a system is impractical in many application since it has to wait until the end of a speech utterance to start the recognition. We therefore put the results of the latter two systems between brackets in Table 2.

To conclude this discussion, we also want to mention some training and execution times. The training of our two-layer 20K reservoir systems takes about 100 hours on a single core 3.0 GHz PC, while recognition takes about two seconds of decoding per second of speech.

## 6 Conclusion and future work

In this paper we showed for the first time that good phoneme recognition on TIMIT can be achieved with a system based on Reservoir Computing. We demonstrated that in order to achieve this, we need large reservoirs (at least 20000 nodes) which are configured in a hierarchical way. By stacking two reservoir layers, we were able to achieve error rates that are competitive with what is attainable using state-of-the-art HMM technology. Our results support the idea that reservoirs can exploit long-term dynamic properties of the articulatory system in continuous speech recognition.

It is acknowledged though that other techniques such as Deep Belief Networks are still outperforming our present system, but the plots and the discussions presented in the course of this paper clearly show a significant margin for further improvement of our system in the near future.

To achieve this improvement we will investigate even larger reservoirs with 50000 and more nodes and we will more thoroughly optimize the parameters of the different reservoirs. Furthermore, we will explore the use of sparsely connected outputs and multi-frame inputs in combination with PCA-based dimensionality reduction. Finally, we will develop an embedded training scheme that permits the training of reservoirs on much larger speech corpora for which only orthographic representations are distributed together with the speech data.

## Acknowledgement

The work presented in this paper is funded by the EC FP7 project ORGANIC (FP7-231267).

## Footnotes

[1]http://reservoir-computing.org/organic/engine

[2]http://www.spraak.org

# References

[1] A. Robinson. An application of recurrent neural nets to phone probability estimation. *IEEE Trans. on Neural Networks*, 5:298–305, 1994.

[2] H. Bourlard and N. Morgan. Continuous speeh recognition by connectionist statistical methods. *IEEE Trans. on Neural Networks*, 4:893–909, 1993.

[3] G. Hinton, S. Osindero, and Y. Teh. A fast learning algorithm for deep belief nets. *Neural Computation*, 18:1527–1554, 2006.

[4] A. Mohamed, G. Dahl, and G. Hinton. Deep belief networks for phone recognition. In *NIPS Workshop on Deep Learning for Speech Recognition and Related Applications*, 2009.

[5] A. Graves and J. Schmidhuber. Framewise phoneme classification with bidirectional LSTM and other neural network architectures. *Neural Networks*, 18:602–610, 2005.

[6] H. Jaeger. Tutorial on training recurrent neural networks, covering BPTT, RTRL, EKF and the echo state network approach (48 pp). Technical report, German National Research Center for Information Technology, 2002.

[7] W. Maass, T. Natschläger, and H. Markram. Real-time computing without stable states: A new framework for neural computation based on perturbations. *Neural Computation*, 14(11):2531–2560, 2002.

[8] D. Verstraeten, B. Schrauwen, M. D'Haene, and D. Stroobandt. An experimental unification of reservoir computing methods. *Neural Networks*, 20:391–403, 2007.

[9] E. Antonelo, B. Schrauwen, and J. Van Campenhout. Generative modeling of autonomous robots and their environments using reservoir computing. *Neural Processing Letters*, 26(3):233–249, 2007.

[10] G. Holzmann and H. Hauser. Echo state networks with filter neurons and a delay & sum readout. *Neural Networks*, 23:244–256, 2010.

[11] D. Verstraeten, B. Schrauwen, and D. Stroobandt. Isolated word recognition using a liquid state machine. In *Proceedings of the 13th European Symposium on Artificial Neural Networks (ESANN)*, pages 435–440, 2005.

[12] M. Skowronski and J. Harris. Automatic speech recognition using a predictive echo state network classifier. *Neural Networks*, 20(3):414–423, 2007.

[13] B. Schrauwen. A hierarchy of recurrent networks for speech recognition. In *NIPS Workshop on Deep Learning for Speech Recognition and Related Applications*, 2009.

[14] J. Garofolo, L. Lamel, W. Fisher, J. Fiscus, D. Pallett, and N. Dahlgren. The DARPA TIMIT acoustic-phonetic continuous speech corpus cd-rom. Technical report, National Institute of Standards and Technology, 1993.

[15] K.F. Lee and H-W. Hon. Speaker-independent phone recognition using hidden markov models. In *IEEE Trans. on Acoustics, Speech and Signal Processing, ASSP*, volume 37, pages 1641–1648, 1989.

[16] H. Jaeger, M. Lukosevicius, D. Popovici, and U. Siewert. Optimization and applications of echo state networks with leaky-integrator neurons. *Neural Networks*, 20:335–352, 2007.

[17] S. Davis and P. Mermelstein. Comparison of parametric representations for monosyllabic word recognition in continuously spoken sentences. *IEEE Trans. on Acoustics Speech & Signal Processing*, 28:357–366, 1980.

[18] L. Van Immerseel and J.P. Martens. Pitch and voiced/unvoiced determination with an auditory model. *Acoustical Society of America*, 91(6):3511–3526, June 1992.

[19] B. Schrauwen, L. Buesing, and R. Legenstein. Computational power and the order-chaos phase transition in reservoir computing. In *Proc. Advances in Neural Information Processing Systems (NIPS)*, volume 21, pages 1425–1432, 2008.

[20] P. Schwarz, P. Matejka, and J. Cernocky. Hierarchical structures of neural networks for phoneme recognition. In *Proc. International Conference on Acoustics, Speech and Signal Processing*, pages 325–328, 2006.

[21] Linguistic Data Consortium. COMLEX english pronunciation lexicon, 2009.

[22] K. Demuynck, J. Roelens, D. Van Compernolle, and P. Wambacq. SPRAAK: An open source speech recognition and automatic annotation kit. In *Procs. Interspeech 2008*, page 495, 2008.

[23] J. Ming and F.J. Smith. Improved phone recognition using bayesian triphone models. *IEEE Trans. on Acoustics, Speech and Signal Processing, ASSP*, 1:409–412, 1998.

